# For valid generalization, the size of the weights is more important than the size of the network

Peter L. Bartlett
Department of Systems Engineering
Research School of Information Sciences and Engineering
Australian National University
Canberra, 0200 Australia
Peter.Bartlett☉anu.edu.au

## Abstract

This paper shows that if a large neural network is used for a pattern classification problem, and the learning algorithm finds a network with small weights that has small squared error on the training patterns, then the generalization performance depends on the size of the weights rather than the number of weights. More specifically, consider an $\ell$-layer feed-forward network of sigmoid units, in which the sum of the magnitudes of the weights associated with each unit is bounded by $A$. The misclassification probability converges to an error estimate (that is closely related to squared error on the training set) at rate $O((cA)^{\ell(\ell+1)/2}\sqrt{(\log n)/m})$ ignoring log factors, where $m$ is the number of training patterns, $n$ is the input dimension, and $c$ is a constant. This may explain the generalization performance of neural networks, particularly when the number of training examples is considerably smaller than the number of weights. It also supports heuristics (such as weight decay and early stopping) that attempt to keep the weights small during training.

## 1 Introduction

Results from statistical learning theory give bounds on the number of training examples that are necessary for satisfactory generalization performance in classification problems, in terms of the Vapnik-Chervonenkis dimension of the class of functions used by the learning system (see, for example, [13, 5]). Baum and Haussler [4] used these results to give sample size bounds for multi-layer threshold networks

that grow at least as quickly as the number of weights (see also [7]). However, for pattern classification applications the VC-bounds seem loose; neural networks often perform successfully with training sets that are considerably smaller than the number of weights. This paper shows that for classification problems on which neural networks perform well, if the weights are not too big, the size of the weights determines the generalization performance.

In contrast with the function classes and algorithms considered in the VC-theory, neural networks used for binary classification problems have real-valued outputs, and learning algorithms typically attempt to minimize the squared error of the network output over a training set. As well as encouraging the correct classification, this tends to push the output away from zero and towards the target values of $\{-1, 1\}$. It is easy to see that if the total squared error of a hypothesis on $m$ examples is no more than $m\epsilon$, then on no more than $m\epsilon/(1-\alpha)^2$ of these examples can the hypothesis have either the incorrect sign or magnitude less than $\alpha$.

The next section gives misclassification probability bounds for hypotheses that are "distinctly correct" in this way on most examples. These bounds are in terms of a scale-sensitive version of the VC-dimension, called the fat-shattering dimension. Section 3 gives bounds on this dimension for feedforward sigmoid networks, which imply the main results. The proofs are sketched in Section 4. Full proofs can be found in the full version [2].

## 2   Notation and bounds on misclassification probability

Denote the space of input patterns by $X$. The space of labels is $\{-1, 1\}$. We assume that there is a probability distribution $P$ on the product space $X \times \{-1, 1\}$, that reflects both the relative frequency of different input patterns and the relative frequency of an expert's classification of those patterns. The learning algorithm uses a class of real-valued functions, called the hypothesis class $H$. An hypothesis $h$ is correct on an example $(x, y)$ if $\text{sgn}(h(x)) = y$, where $\text{sgn}(\alpha) : \mathbb{R} \to \{-1, 1\}$ takes value 1 iff $\alpha \geq 0$, so the misclassification probability (or error) of $h$ is defined as

$$\text{er}_P(h) = P\{(x, y) \in X \times \{-1, 1\} : \text{sgn}(h(x)) \neq y\}.$$

The crucial quantity determining misclassification probability is the fat-shattering dimension of the hypothesis class $H$. We say that a sequence $x_1, \ldots, x_d$ of $d$ points from $X$ is shattered by $H$ if functions in $H$ can give all classifications of the sequence. That is, for all $b = (b_1, \ldots, b_m) \in \{-1, 1\}^m$ there is an $h$ in $H$ satisfying $\text{sgn}(h(x_i)) = b_i$. The VC-dimension of $H$ is defined as the size of the largest shattered sequence.[1]

For a given scale parameter $\gamma > 0$, we say that a sequence $x_1, \ldots, x_d$ of $d$ points from $X$ is $\gamma$-shattered by $H$ if there is a sequence $r_1, \ldots, r_d$ of real values such that for all $b = (b_1, \ldots, b_m) \in \{-1, 1\}^m$ there is an $h$ in $H$ satisfying $(h(x_i) - r_i)b_i \geq \gamma$. The fat-shattering dimension of $H$ at $\gamma$, denoted $\text{fat}_H(\gamma)$, is the size of the largest $\gamma$-shattered sequence. This dimension reflects the complexity of the functions in the class $H$, when examined at scale $\gamma$. Notice that $\text{fat}_H(\gamma)$ is a nonincreasing function of $\gamma$. The following theorem gives generalization error bounds in terms of $\text{fat}_H(\gamma)$. A related result, that applies to the case of no errors on the training set, will appear in [12].

**Theorem 1** *Define the input space $X$, hypothesis class $H$, and probability distribution $P$ on $X \times \{-1, 1\}$ as above. Let $0 < \delta < 1/2$, and $0 < \gamma < 1$. Then, with probability $1 - \delta$ over the training sequence $(x_1, y_1), \ldots, (x_m, y_m)$ of $m$ labelled*

*examples, every hypothesis h in H satisfies*

$$\mathrm{er}_P(h) < \frac{1}{m} \left| \{i : |h(x_i)| < \gamma \ or \ \mathrm{sgn}(h(x_i)) \neq y_i\} \right| + \epsilon(\gamma, m, \delta),$$

*where*

$$\epsilon^2(\gamma, m, \delta) = \frac{2}{m} \left( d \ln(50em/d) \log_2(1250m) + \ln(4/\delta) \right), \tag{1}$$

*and* $d = \mathrm{fat}_H(\gamma/16)$.

### 2.1   Comments

It is informative to compare this result with the standard VC-bound. In that case, the bound on misclassification probability is

$$\mathrm{er}_P(h) < \frac{1}{m} \left| \{i : \mathrm{sgn}(h(x_i)) \neq y_i\} \right| + \left( \frac{c}{m} \left( d \log(m/d) + \log(1/\delta) \right) \right)^{1/2},$$

where $d = \mathrm{VCdim}(H)$ and $c$ is a constant. We shall see in the next section that there are function classes $H$ for which $\mathrm{VCdim}(H)$ is infinite but $\mathrm{fat}_H(\gamma)$ is finite for all $\gamma > 0$; an example is the class of functions computed by any two-layer neural network with an arbitrary number of parameters but constraints on the size of the parameters. It is known that if the learning algorithm and error estimates are constrained to make use of the sample only by considering the proportion of training examples that hypotheses misclassify, there are distributions $P$ for which the second term in the VC-bound above cannot be improved by more than log factors. Theorem 1 shows that it can be improved if the learning algorithm makes use of the sample by considering the proportion of training examples that are correctly classified and have $|h(x_i)| < \gamma$. It is possible to give a lower bound (see the full paper [2]) which, for the function classes considered here, shows that Theorem 1 also cannot be improved by more than log factors.

The idea of using the magnitudes of the values of $h(x_i)$ to give a more precise estimate of the generalization performance was first proposed by Vapnik in [13] (and was further developed by Vapnik and co-workers). There it was used only for the case of linear hypothesis classes. Results in [13] give bounds on misclassification probability for a test sample, in terms of values of $h$ on the training and test data. This result is extended in [11], to give bounds on misclassification probability (that is, for unseen data) in terms of the values of $h$ on the training examples. This is further extended in [12] to more general function classes, to give error bounds that are applicable when there is a hypothesis with no errors on the training examples. Lugosi and Pintér [9] have also obtained bounds on misclassification probability in terms of similar properties of the class of functions containing the true regression function (conditional expectation of $y$ given $x$). However, their results do not extend to the case when the true regression function is not in the class of real-valued functions used by the estimator.

It seems unnatural that the quantity $\gamma$ is specified in advance in Theorem 1, since it depends on the examples. The full paper [2] gives a similar result in which the statement is made uniform over all values of this quantity.

## 3   The fat-shattering dimension of neural networks

Bounds on the VC-dimension of various neural network classes have been established (see [10] for a review), but these are all at least linear in the number of parameters. In this section, we give bounds on the fat-shattering dimension for several neural network classes.

We assume that the input space $X$ is some subset of $\mathbb{R}^n$. Define a sigmoid unit as a function from $\mathbb{R}^k$ to $\mathbb{R}$, parametrized by a vector of weights $w \in \mathbb{R}^k$. The unit computes $x \mapsto \sigma(x \cdot w)$, where $\sigma$ is a fixed bounded function satisfying a Lipchitz condition. (For simplicity, we ignore the offset parameter. It is equivalent to including an extra input with a constant value.) A multi-layer feed-forward sigmoid network of depth $\ell$ is a network of sigmoid units with a single output unit, which can be arranged in a layered structure with $\ell$ layers, so that the output of a unit passes only to the inputs of units in later layers. We will consider networks in which the weights are bounded. The relevant norm is the $\ell_1$ norm: for a vector $w \in \mathbb{R}^k$, define $\|w\|_1 = \sum_{i=1}^k |w_i|$. The following result gives a bound on the fat-shattering dimension of a (bounded) linear combination of real-valued functions, in terms of the fat-shattering dimension of the basis function class. We can apply this result in a recursive fashion to give bounds for single output feed-forward networks.

**Theorem 2** *Let $F$ be a class of functions that map from $X$ to $[-M/2, M/2]$, such that $0 \in F$ and, for all $f$ in $F$, $-f \in F$. For $A > 0$, define the class $H$ of weight-bounded linear combinations of functions from $F$ as*

$$H = \left\{ \sum_{i=1}^k w_i f_i : k \in \mathbb{N}, f_i \in F, \|w\|_1 \le A \right\}.$$

*Suppose $\gamma > 0$ is such that $d = \mathrm{fat}_F(\gamma/(32A)) \ge 1$. Then $\mathrm{fat}_H(\gamma) \le (cM^2A^2d/\gamma^2)\log^2(MAd/\gamma)$, for some constant $c$.*

Gurvits and Koiran [6] have shown that the fat-shattering dimension of the class of two-layer networks with bounded output weights and linear threshold hidden units is $O\left((A^2n^2/\gamma^2)\log(n/\gamma)\right)$, when $X = \mathbb{R}^n$. As a special case, Theorem 2 improves this result.

Notice that the fat-shattering dimension of a function class is not changed by more than a constant factor if we compose the functions with a fixed function satisfying a Lipschitz condition (like the standard sigmoid function). Also, for $X = \mathbb{R}^n$ and $H = \{x \mapsto x_i\}$ we have $\mathrm{fat}_H(\gamma) \le \log n$ for all $\gamma$. Finally, for $H = \{x \mapsto w \cdot x : w \in \mathbb{R}^n\}$ we have $\mathrm{fat}_H(\gamma) \le n$ for all $\gamma$. These observations, together with Theorem 2, give the following corollary. The $\tilde{O}(\cdot)$ notation suppresses log factors. (Formally, $f = \tilde{O}(g)$ if $f = o(g^{1+\alpha})$ for all $\alpha > 0$.)

**Corollary 3** *If $X \subseteq \mathbb{R}^n$ and $H$ is the class of two-layer sigmoid networks with the weights in the output unit satisfying $\|w\|_1 \le A$, then $\mathrm{fat}_H(\gamma) = \tilde{O}\left(A^2n/\gamma^2\right)$.*

*If $X = \{x \in \mathbb{R}^n : \|x\|_\infty \le B\}$ and the hidden unit weights are also bounded, then $\mathrm{fat}_H(\gamma) = \tilde{O}\left(B^2A^6(\log n)/\gamma^4\right)$.*

Applying Theorem 2 to this result gives the following result for deeper networks. Notice that there is no constraint on the number of hidden units in any layer, only on the total magnitude of the weights associated with a processing unit.

**Corollary 4** *For some constant $c$, if $X \subseteq \mathbb{R}^n$ and $H$ is the class of depth $\ell$ sigmoid networks in which the weight vector $w$ associated with each unit beyond the first layer satisfies $\|w\|_1 \le A$, then $\mathrm{fat}_H(\gamma) = \tilde{O}\left(n(cA)^{\ell(\ell-1)}/\gamma^{2(\ell-1)}\right)$.*

*If $X = \{x \in \mathbb{R}^n : \|x\|_\infty \le B\}$ and the weights in the first layer units also satisfy $\|w\|_1 \le A$, then $\mathrm{fat}_H(\gamma) = \tilde{O}\left(B^2(cA)^{\ell(\ell+1)}/\gamma^{2\ell}\log n\right)$.*

In the first part of this corollary, the network has fat-shattering dimension similar to the VC-dimension of a linear network. This formalizes the intuition that when the weights are small, the network operates in the "linear part" of the sigmoid, and so behaves like a linear network.

## 3.1 Comments

Consider a depth $\ell$ sigmoid network with bounded weights. The last corollary and Theorem 1 imply that if the training sample size grows roughly as $B^2 A^{\ell^2}/\epsilon^2$, then the misclassification probability of a network is within $\epsilon$ of the proportion of training examples that the network classifies as "distinctly correct."

These results give a plausible explanation for the generalization performance of neural networks. If, in applications, networks with many units have small weights and small squared error on the training examples, then the VC-dimension (and hence number of parameters) is not as important as the magnitude of the weights for generalization performance.

It is possible to give a version of Theorem 1 in which the probability bound is uniform over all values of a complexity parameter indexing the function classes (using the same technique mentioned at the end of Section 2.1). For the case of sigmoid network classes, indexed by a weight bound, minimizing the resulting bound on misclassification probability is equivalent to minimizing the sum of a sample error term and a penalty term involving the weight bound. This supports the use of two popular heuristic techniques, weight decay and early stopping (see, for example, [8]), which aim to minimize squared error while maintaining small weights.

These techniques give bounds on the fat-shattering dimension and hence generalization performance for any function class that can be expressed as a bounded number of compositions of either bounded-weight linear combinations or scalar Lipschitz functions with functions in a class that has finite fat-shattering dimension. This includes, for example, radial basis function networks.

## 4 Proofs

### 4.1 Proof sketch of Theorem 1

For $A \subseteq S$, where $(S, \rho)$ is a pseudometric space, a set $T \subseteq S$ is an $\epsilon$-cover of $A$ if for all $a$ in $A$ there is a $t$ in $T$ with $\rho(t, a) < \epsilon$. We define $\mathcal{N}(A, \epsilon, \rho)$ as the size of the smallest $\epsilon$-cover of $A$. For $x = (x_1, \ldots, x_m) \in X^m$, define the pseudometric $d_{\ell_\infty(x)}$ on the set $S$ of functions defined on $X$ by $d_{\ell_\infty(x)}(f, g) = \max_i |f(x_i) - g(x_i)|$. For a set $A$ of functions, denote $\max_{x \in X^m} \mathcal{N}(A, \epsilon, d_{\ell_\infty(x)})$ by $\mathcal{N}_\infty(A, \epsilon, m)$. Alon *et al.* [1] have obtained the following bound on $\mathcal{N}_\infty$ in terms of the fat-shattering dimension.

**Lemma 5** *For a class $F$ of functions that map from $\{1, \ldots, n\}$ to $\{1, \ldots, b\}$ with* $\mathrm{fat}_F(1) \leq d$, $\log_2 \mathcal{N}_\infty(F, 2, n) < 1 + \log_2(nb^2) \log_2 \left( \sum_{i=0}^d \binom{n}{i} b^i \right)$, *provided that* $n \geq 1 + \log_2 \left( \sum_{i=0}^d \binom{n}{i} b^i \right)$.

For $\gamma > 0$ define $\pi_\gamma : \mathbb{R} \to [-\gamma, \gamma]$ as the piecewise-linear squashing function satisfying $\pi_\gamma(\alpha) = \gamma$ if $\alpha \geq \gamma$, $\pi_\gamma(\alpha) = -\gamma$ if $\alpha \leq -\gamma$, and $\pi_\gamma(\alpha) = \alpha$ otherwise. For a class $H$ of real-valued functions, define $\pi_\gamma(H)$ as the set of compositions of $\pi_\gamma$ with functions in $H$.

**Lemma 6** *For $X$, $H$, $P$, $\delta$, and $\gamma$ as in Theorem 1,*

$$P^m \left\{ z : \exists h \in H, \mathrm{er}_P(h) \geq \left( \frac{2}{m} \ln \left( \frac{2\mathcal{N}_\infty(\pi_\gamma(H), \gamma/2, 2m)}{\delta} \right) \right)^{1/2} + \frac{1}{m} |\{i : |h(x_i)| < \gamma \text{ or } \mathrm{sgn}(h(x_i)) \neq y_i\}| \right\} < \delta.$$

The proof of the lemma relies on the observation that

$$P^m \left\{ z : \exists h \in H, \operatorname{er}_P(h) \geq \epsilon + \frac{1}{m} |\{i : |h(x_i)| < \gamma \text{ or } \operatorname{sgn}(h(x_i)) \neq y_i\}| \right\}$$

$$\leq P^m \left\{ z : \exists h \in H, P\left(|\pi_\gamma(h(x)) - \gamma y| \geq \gamma\right) \geq \epsilon + \frac{1}{m} |\{i : \pi_\gamma(h(x_i)) \neq \gamma y_i\}| \right\}.$$

We then use a standard symmetrization argument and the permutation argument introduced by Vapnik and Chervonenkis to bound this probability by the probability under a random permutation of a double-length sample that a related property holds. For any fixed sample, we can then use Pollard's approach of approximating the hypothesis class using a $(\gamma/2)$-cover, except that in this case the appropriate cover is with respect to the $\ell_\infty$ pseudometric. Applying Hoeffding's inequality gives the lemma.

To prove Theorem 1, we need to bound the covering numbers in terms of the fat-shattering dimension. It is easy to apply Lemma 5 to a quantized version of the function class to get such a bound, taking advantage of the range constraint imposed by the squashing function $\pi_\gamma$.

## 4.2 Proof sketch of Theorem 2

For $x = (x_1, \ldots, x_m) \in X^m$, define the pseudometric $d_{\ell_1(x)}$ on the class of functions defined on $X$ by $d_{\ell_1(x)}(f,g) = \frac{1}{m}\sum_{i=1}^m |f(x_i) - g(x_i)|$. Similarly, define $d_{\ell_2(x)}(f,g) = \left(\frac{1}{m}\sum_{i=1}^m (f(x_i)-g(x_i))^2\right)^{1/2}$. If $A$ is a set of functions defined on $X$, denote $\max_{x \in X^m} \mathcal{N}(A, \gamma, d_{\ell_1(x)})$ by $\mathcal{N}_1(A, \gamma, m)$, and similarly for $\mathcal{N}_2(A, \gamma, m)$.

The idea of the proof of Theorem 2 is to first derive a general upper bound on an $\ell_1$ covering number of the class $H$, and then apply the following result (which is implicit in the proof of Theorem 2 in [3]) to give a bound on the fat-shattering dimension.

**Lemma 7** *For a class $F$ of $[0,1]$-valued functions on $X$ satisfying $\operatorname{fat}_F(4\gamma) \geq d$, we have $\log_2 \mathcal{N}_1(F, \gamma, d) \geq d/32$.*

To derive an upper bound on $\mathcal{N}_1(\gamma, H, m)$, we start with the bound that Lemma 5 implies on the $\ell_\infty$ covering number $\mathcal{N}_\infty(F, \gamma, m)$ for the class $F$ of hidden unit functions. Since $d_{\ell_2}(f,g) \leq d_{l_\infty}(f,g)$, this implies the following bound on the $\ell_2$ covering number for $F$ (provided $m$ satisfies the condition required by Lemma 5, and it turns out that the theorem is trivial otherwise).

$$\log_2 \mathcal{N}_2(F, \gamma, m) < 1 + d\log_2\left(\frac{4emM}{d\gamma}\right)\log_2\left(\frac{9mM^2}{\gamma^2}\right). \tag{2}$$

Next, we use the following result on approximation in $\ell_2$, which A. Barron attributes to Maurey.

**Lemma 8 (Maurey)** *Suppose $G$ is a Hilbert space and $F \subseteq G$ has $\|f\| \leq b$ for all $f$ in $F$. Let $f$ be an element from the convex closure of $F$. Then for all $k \geq 1$ and all $c > b^2 - \|f\|^2$, there are functions $\{f_1, \ldots, f_k\} \subseteq F$ such that $\left\|f - \frac{1}{k}\sum_{i=1}^k f_i\right\|^2 \leq \frac{c}{k}$.*

This implies that any element of $H$ can be approximated to a particular accuracy (with respect to $\ell_2$) using a fixed linear combination of a small number of elements of $F$. It follows that we can construct an $\ell_2$ cover of $H$ from the $\ell_2$ cover of $F$; using Lemma 8 and Inequality 2 shows that

$$\log_2 \mathcal{N}_2(H, \gamma, m) < \frac{2M^2A^2}{\gamma^2}\left(1 + d\log_2\left(\frac{8emMA}{d\gamma}\right)\log_2\left(\frac{36mM^2A^2}{\gamma^2}\right)\right).$$

Now, Jensen's inequality implies that $d_{l_1(x)}(f,g) \leq d_{l_2(x)}(f,g)$, which gives a bound on $\mathcal{N}_1(H,\gamma,m)$. Comparing with the lower bound given by Lemma 7 and solving for $m$ gives the result. A more refined analysis for the neural network case involves bounding $\mathcal{N}_2$ for successive layers, and solving to give a bound on the fat-shattering dimension of the network.

## Acknowledgements

Thanks to Andrew Barron, Jonathan Baxter, Mike Jordan, Adam Kowalczyk, Wee Sun Lee, Phil Long, John Shawe-Taylor, and Robert Slaviero for helpful discussions and comments.

## Footnotes

[1] In fact, according to the usual definition, this is the VC-dimension of the class of thresholded versions of functions in $H$.

## References

[1] N. Alon, S. Ben-David, N. Cesa-Bianchi, and D. Haussler. Scale-sensitive dimensions, uniform convergence, and learnability. In *Proceedings of the 1993 IEEE Symposium on Foundations of Computer Science.* IEEE Press, 1993.

[2] P. L. Bartlett. The sample complexity of pattern classification with neural networks: the size of the weights is more important than the size of the network. Technical report, Department of Systems Engineering, Australian National University, 1996. (available by anonymous ftp from `syseng.anu.edu.au:pub/peter/TR96d.ps`).

[3] P. L. Bartlett, S. R. Kulkarni, and S. E. Posner. Covering numbers for real-valued function classes. Technical report, Australian National University and Princeton University, 1996.

[4] E. Baum and D. Haussler. What size net gives valid generalization? *Neural Computation*, 1(1):151–160, 1989.

[5] A. Blumer, A. Ehrenfeucht, D. Haussler, and M. K. Warmuth. Learnability and the Vapnik-Chervonenkis dimension. *J. ACM*, 36(4):929–965, 1989.

[6] L. Gurvits and P. Koiran. Approximation and learning of convex superpositions. In *Computational Learning Theory: EUROCOLT'95*, 1995.

[7] D. Haussler. Decision theoretic generalizations of the PAC model for neural net and other learning applications. *Inform. Comput.*, 100(1):78–150, 1992.

[8] J. Hertz, A. Krogh, and R. G. Palmer. *Introduction to the Theory of Neural Computation*. Addison-Wesley, 1991.

[9] G. Lugosi and M. Pintér. A data-dependent skeleton estimate for learning. In *Proc. 9th Annu. Conference on Comput. Learning Theory*. ACM Press, New York, NY, 1996.

[10] W. Maass. Vapnik-Chervonenkis dimension of neural nets. In M. A. Arbib, editor, *The Handbook of Brain Theory and Neural Networks*, pages 1000–1003. MIT Press, Cambridge, 1995.

[11] J. Shawe-Taylor, P. L. Bartlett, R. C. Williamson, and M. Anthony. A framework for structural risk minimisation. In *Proc. 9th Annu. Conference on Comput. Learning Theory*. ACM Press, New York, NY, 1996.

[12] J. Shawe-Taylor, P. L. Bartlett, R. C. Williamson, and M. Anthony. Structural risk minimization over data-dependent hierarchies. Technical report, 1996.

[13] V. N. Vapnik. *Estimation of Dependences Based on Empirical Data*. Springer-Verlag, New York, 1982.